# Subspace-Based Face Recognition in Analog VLSI

**Gonzalo Carvajal, Waldo Valenzuela and Miguel Figueroa**
Department of Electrical Engineering, Universidad de Concepción
Casilla 160-C, Correo 3, Concepción, Chile
*{gcarvaja, waldovalenzuela, miguel.figueroa}@udec.cl*

## Abstract

We describe an analog-VLSI neural network for face recognition based on subspace methods. The system uses a dimensionality-reduction network whose coefficients can be either programmed or learned on-chip to perform PCA, or programmed to perform LDA. A second network with user-programmed coefficients performs classification with Manhattan distances. The system uses on-chip compensation techniques to reduce the effects of device mismatch. Using the ORL database with 12x12-pixel images, our circuit achieves up to 85% classification performance (98% of an equivalent software implementation).

## 1  Introduction

Subspace-based techniques for face recognition, such as Eigenfaces [1] and Fisherfaces [2], take advantage of the large redundancy present in most images to compute a lower-dimensional representation of their input data and stored patterns, and perform classification in the reduced subspace. Doing so substantially lowers the storage and computational requirements of the face-recognition task.

However, most techniques for dimensionality reduction require a high computational throughput to transform images from the large input data space to the feature subspace. Therefore, software [3] even dedicated digital hardware implementations [4, 5] are too large and power-hungry to be used in highly portable systems. Analog VLSI circuits can compute using orders of magnitude less power and die area than their digital counterparts, but their performance is limited by signal offsets, parameter mismatch, charge leakage and nonlinear behavior, particularly in large-scale systems. Traditional circuit-design techniques can reduce these effects, but they increase power and area, rendering analog solutions less attractive.

In this paper, we present a neural network for face recognition which implements Principal Components Analysis (PCA) and Linear Discriminant Analysis (LDA) for dimensionality reduction, and Manhattan distances and a loser-take-all (LTA) circuit for classification. We can download the network weights in a chip-in-the loop configuration, or use on-chip learning to compute PCA coefficients. We use local adaptation to achieve good classification performance in the presence of device mismatch. The circuit die area is 2.2mm$^2$ in a 0.35$\mu$m CMOS process, with an estimated power dissipation of 18mW. Using PCA reduction and a hard classifier, our network achieves up to 83% accuracy on the Olivetti Research Labs (ORL) face database [6] using 12x12-pixel images, which corresponds to 99% of the accuracy of a software implementation of the algorithm. Using LDA projections and a software Radial Basis Function (RBF) network on the hardware-computed distances yields 85% accuracy (98% of the software performance).

## 2 Eigenspace based face recognition methods

The problem of face recognition consists of assigning an identity to an unknown face by comparing it to a database of labeled faces. However, the dimensionality of the input images is usually so high that performing the classification on the original data becomes prohibitively expensive.

Fortunately, human faces exhibit relatively regular statistics; therefore, their intrinsic dimensionality is much lower than that of their images. Subspace methods transform the input images to reduce their dimensionality, and perform the classification task on this lower-dimensional feature space. In particular, the Eigenfaces [1] method performs dimensionality reduction using PCA, and classification by choosing the stored face with the lowest distance to the input data.

Principal Components Analysis uses a linear transformation from the input space to the feature space, which preserves most of the information (in the mean-square error sense) present in the original vector. Consider a column vector $\mathbf{x}$ of dimension $n$, formed by the concatenated columns of the input image. Let the matrix $\mathbf{X}_{n \times N} = \{\mathbf{x}_1, \mathbf{x}_2, \ldots, \mathbf{x}_N\}$ represent a set of $N$ images, such as the image database available for a face recognition task. PCA computes a new matrix $\mathbf{Y}_{m \times N}$, with $m < n$:

$$\mathbf{Y} = \mathbf{W}^{*\mathrm{T}} \mathbf{X} \tag{1}$$

The columns of $\mathbf{Y}$ are the lower-dimensional projections of the original images in the feature space. The columns of the orthogonal transformation matrix $\mathbf{W}^*$ are the eigenvectors associated to the $m$ largest eigenvalues of the covariance matrix of the original image space.

Upon presentation of a new face image, the Eigenfaces method first transforms this image into the feature space using the transformation matrix $\mathbf{W}^*$, and then computes the distance between the reduced image and each image class in the reference database. The image is classified with the identity of the closest reference pattern.

Fisherfaces [2] performs dimensionality reduction using Linear Discriminant Analysis (LDA). LDA takes advantage of labeled data to maximize the distance between classes in the projected subspace. Considering $\mathbf{X}_c$ , $c = 1, \ldots, N_c$ as subsets of $\mathbf{X}$ containing $N_i$ images of the same subject, LDA defines two matrices:

$$\mathbf{S}_W = \sum_{i=1}^{c} \sum_{\mathbf{x}_k \in \mathbf{X}_c} (\mathbf{x}_k - \mathbf{m}_i)(\mathbf{x}_k - \mathbf{m}_i)^{\mathrm{T}} \text{ , with } \mathbf{m}_i = \frac{1}{N_i} \sum_{k=1}^{N_i} \mathbf{x}_k \tag{2}$$

$$\mathbf{S}_B = \sum_{i=1}^{c} N_i (\mathbf{m}_i - \mathbf{m})(\mathbf{m}_i - \mathbf{m})^{\mathrm{T}} \tag{3}$$

where $\mathbf{S}_W$ represents the scatter (variance) within classes, and $\mathbf{S}_B$ is the scatter between different classes. To perform the dimensionality reduction of Eqn. (1), LDA constructs $\mathbf{W}^*$ such that its columns are the $m$ largest eigenvectors of $\mathbf{S}_W^{-1}\mathbf{S}_B$. This requires $\mathbf{S}_W$ to be non-singular, which is often not the case; therefore, LDA frequently uses a PCA preprocessing stage [2].

Fisherfaces can perform classification using a hard classifier on the computed distances between the test data and stored patterns in the LDA subspace, as in Eigenfaces, or it can use a Radial Basis Function (RBF) network. RBF uses a hidden layer of neurons with Gaussian activation functions to detect clusters in the projected subspace.

Traditionally, the subspace method use Euclidian distances. However, our experiments show that, as long as the dimensionality reduction preserves enough distance between classes, less computationally expensive distance metrics such as Manhattan distance are equally effective for classification. The Manhattan distance between two vectors $\mathbf{x} = [x_1 \ \ldots \ x_n]$ and $\mathbf{y} = [y_1 \ \ldots \ y_n]$ is given by:

$$d = \sum_{i=1}^{n} |x_i - y_i| \tag{4}$$

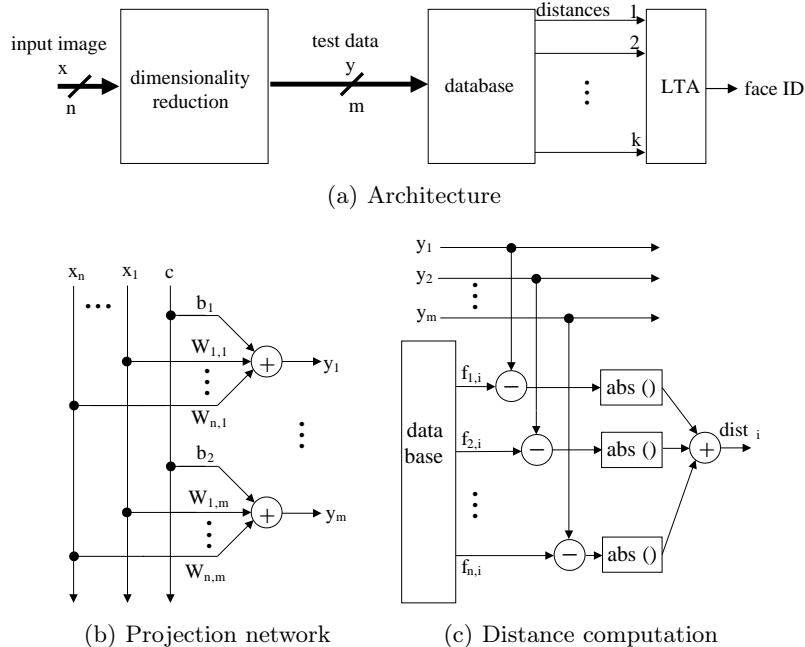

(a) Architecture

(b) Projection network

(c) Distance computation

Figure 1: Face-recognition hardware. (a) Architecture. A dimensionality-reduction network projects a $n$-dimensional image onto $m$ dimensions, and loser-take-all (LTA) circuit labels the image by choosing the nearest stored face in the reduced space. (b) The dimensionality reduction network is an array of linear combiners with weights that have been pre-computed or learned on chip. (c) The distance circuit computes the Manhattan distance between the $m$ projections of the test image and the stored face database. In our current implementation, $n = 144$, $m = 39$, and $k = 40$.

## 3   Hardware Implementation

Fig. 1(a) shows the architecture of our face-recognition network. It follows the signal flow described in Section 2, where the $n$-dimensional test image $\mathbf{x}$ is first projected onto the $m$-dimensional feature space (test data $\mathbf{y}$) using an array of $m$ $n$-input analog linear combiners, shown in Fig. 1(b). The constant input $c$ is a bias used to compensate for the offset introduced by the analog multipliers. The network also stores the $m$ projections of the database face set (the training set) in an array of analog memories. A distance computation block, shown in Fig. 1(c), computes the Manhattan distance between each labeled element in the stored training set and the reduced test data $\mathbf{y}$. A loser-take-all (LTA) circuit, currently implemented in software, selects the smallest distance and labels the test image with the selected class.

The linear combiners are based on the synapse shown in Fig. 2(a). An analog Gilbert multiplier computes the product of each pixel of the input image, represented as a differential voltage, and the local synaptic weight. An accurate transformation requires a multiplier response that is linear in the pixel value, therefore we designed the multipliers to maximize the linearity of that input. Device mismatch introduces offsets and gain variance across different multipliers in the network; we describe the calibration techniques used to compensate for these effects in Section 4. The multipliers provide a differential current output, therefore we can add them across a single neuron by connecting them to common wires.

Each synaptic weight is stored in an analog nonvolatile memory cell [7] based on floating-gate transistors, shown also in Fig. 2(a). The cell features linear weight-updates based on digital pulses applied to the terminals *inc* and *dec*. Using local calibration, also based on floating gates, we independently tune each synapse to achieve symmetric updates in the

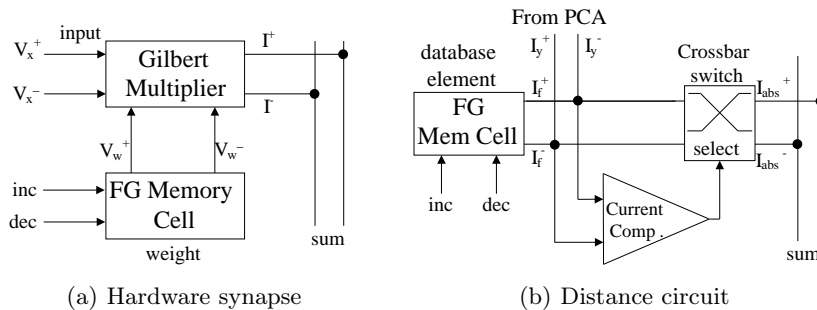

(a) Hardware synapse    (b) Distance circuit

Figure 2: (a) The synapse is comprised by a Gilbert multiplier and a nonvolatile analog memory cell with local calibration. The output currents are summed across each neuron. (b) Each component of the Manhattan distance is computed as the subtraction of the corresponding principal components and an optional inversion based on the sign of the result. The output currents are summed across all components.

presence of device mismatch, and to make the update rates uniform across the entire chip. As a result, the resolution of the memory cell exceeds 12 bits in a $0.35\mu$m CMOS process.

Fig. 2(b) depicts the circuit used to compute the Manhattan distance between the test data and the stored patterns. Each projection of the training set is stored as a current in an analog memory cell, simpler and smaller than the cell used in the dimensionality reduction network, and written using a self-limiting write process. The difference between each projection of the pattern and the test input is computed by inverting the polarity of one of the signals and adding the currents. To compute the absolute value, a current comparator based on a simple transconductance amplifier determines the sign of the result and uses a $2 \times 2$ crossbar switch to invert the polarity of the outputs if needed.

As stated in Section 5, our current implementation considers 12×12-pixel images ($n = 144$ in Fig. 1). We compute 39 projections using PCA and LDA, and perform the classification using 40 Manhattan-distance units on the 39-dimensional projections. The next section analyzes the effects of device mismatch on the dimensionality-reduction network.

## 4 Analog implementation of dimensionality reduction networks

The arithmetic distortions introduced by the nonlinear transfer function of the analog multipliers, coupled with the effects of device mismatch (offsets and gains), affect the accuracy of the operations performed by the reduction network and become the limiting factor in the classification performance. In order to achieve good performance, we must calibrate the network to compensate for the effect of these limitations.

In this section, we analyze and design solutions for two different cases. First, we consider the case when a computer performs PCA or LDA to determine $\mathbf{W}^*$ off-line, and downloads the weights onto the chip. Second, we analyze the performance of adaptive on-chip computation of PCA using a Hebbian-learning algorithm. In both cases, we design mechanisms that use local on-chip adaptation to compensate for the offsets and gain variances introduced by device mismatch, thus improving classification performance. In the following analysis we assume that the inputs have zero mean and have been normalized. Also, for simplicity, we assume that the inputs and weights are operating within the linear range of the multipliers. We remove these assumptions when presenting experimental results. Thus, our analysis uses a simplified model of the analog multipliers given by:

$$o = (a_x x + \gamma_x)(a_w w + \gamma_w) \tag{5}$$

where $o$ is the multiplier output, $x$ and $w$ are the inputs, $\gamma_x$ and $\gamma_w$ represent the input offsets, and $a_x$ and $a_w$ are the multiplier gains associated with each input. These parameters vary across different multipliers due to device mismatch and are unknown at design time, and difficult to determine even after circuit fabrication.

## 4.1 Dimensionality reduction with precomputed weights

Let us consider an analog linear combiner such as the one depicted in Fig. 1(b), which computes the first projection $y$ of $\mathbf{x}$, using the first column $\mathbf{w}^*$ of the software precomputed optimal transformation $\mathbf{W}^*$ of Eqn. (1). Using the simplified multiplier linear model of Eqn. (5), the linear combiner computes the first projection as:

$$\overline{y} = \mathbf{x}^{\mathrm{T}}(\mathbf{A_x}\mathbf{A_w}\mathbf{w}^* + \mathbf{A_x}\boldsymbol{\gamma_w}) + \boldsymbol{\gamma_x}^{\mathrm{T}}(\mathbf{A_w}\mathbf{w}^* + \boldsymbol{\gamma_w}) \tag{6}$$

where $\mathbf{A_x} = \mathrm{diag}([a_{x_1} \ \ldots \ a_{x_n}])$, $\mathbf{A_w} = \mathrm{diag}([a_{w_1} \ \ldots \ a_{w_n}])$, $\boldsymbol{\gamma_x} = [\gamma_{x_1} \ \ldots \ \gamma_{x_n}]^{\mathrm{T}}$, and $\boldsymbol{\gamma_w} = [\gamma_{w_1} \ \ldots \ \gamma_{w_n}]^{\mathrm{T}}$ represent the gains and offsets of each multiplier. Eqn. (6) shows that device mismatch has two effects on the output: the first term modifies the effective weight value of the network, and the second term represents an offset added to the output ($\mathbf{w}^*$ is a constant).

Replacing $\mathbf{w}^*$ with an adaptive version $\mathbf{w}_k$, the structure becomes a classic adaptive linear combiner which, using the optimal weights to generate a reference output signal, can be trained using the well known Least-Mean Squares (LMS) algorithm. Adding a bias synapse $b$ with constant input $c$ and training the network with LMS, the weights converge to [7]:

$$\overline{\mathbf{w}}^* = (\mathbf{A_x}\mathbf{A_w})^{-1}(\mathbf{w}^* - \mathbf{A_x}\boldsymbol{\gamma_w}) \tag{7}$$

$$b^* = -(\boldsymbol{\gamma_x}^{\mathrm{T}}(\mathbf{A_w}\overline{\mathbf{w}}^* + \boldsymbol{\gamma_w}) + c\gamma_b)(ca_b)^{-1} \tag{8}$$

where $a_b$ and $\gamma_b$ are the gain and offset of the analog multiplier associated to the bias input $c$. These weight values fully compensate for the effects of gain mismatch and offsets.

In our hardware implementation, we use $m$ adaptive linear combiners to compute every projection in the feature space, and calibrate these circuits using on-chip LMS local adaptation to compute and store the optimal weight values of Eqns. (7) and (8), achieving a good approximation of the optimal output $\mathbf{Y}$. Fig. 3(a) shows our analog-VLSI implementation of LMS. We train the weight values in the memory cells by providing inputs and a reference output to each linear combiner, and use an on-chip pulse-based compact implementation of the LMS learning rule. In order to improve the convergence of the algorithm, we draw the inputs from a zero-mean random Gaussian distribution. Thus, the performance of the dimensionality reduction network is ultimately limited by the resolution of the memory cells, the reference noise, the learning rate of the LMS training stage and linearity of the multipliers. This last effect can be controlled by restricting the dynamic range of the input to linear range of the multipliers.

To measure the accuracy of our implementation, we computed (in software) the first 10 principal components of one half the Olivetti Research Labs (ORL) face database, reduced to 12x12 pixels, and used our on-chip implementation of LMS to train the hardware network to learn the coefficients. We then measured the output of the circuit on the other half of the database. Fig. 3(b) plots the RMS value of the error between the circuit output and the software results, normalized to the RMS value of each principal component. The figure also shows the error when we wrote the coefficients onto the circuit in open-loop, without using LMS. In this case, offset and gain mismatch completely obscure the information present in the signal. LMS training compensates for these effects, and reduces the error energy to between 0.25% and 1% of the energy of the signal. A different experiment (not shown) computing LDA coefficients yields equivalent results.

## 4.2 On-chip PCA computation

In some cases, such as when the face-recognition network is integrated with a camera on a single chip, it may be necessary to train the face database on-chip. It is not practical for the chip to include the hardware resources to compute the optimal weights from the eigenvalue analysis of the training set's covariance matrix, therefore we compute them on chip using the standard Generalized Hebbian Algorithm (GHA). The computation of the first principal component and the learning rule to update the weights at time $k$ are:

$$y_k = \mathbf{x}_k^T \mathbf{w}_k \tag{9}$$

$$\Delta\mathbf{w}_k = \mu y_k(\mathbf{x}_k - \mathbf{x}'_k) \tag{10}$$

$$\mathbf{x}'_k = y_k\mathbf{w}_k \tag{11}$$

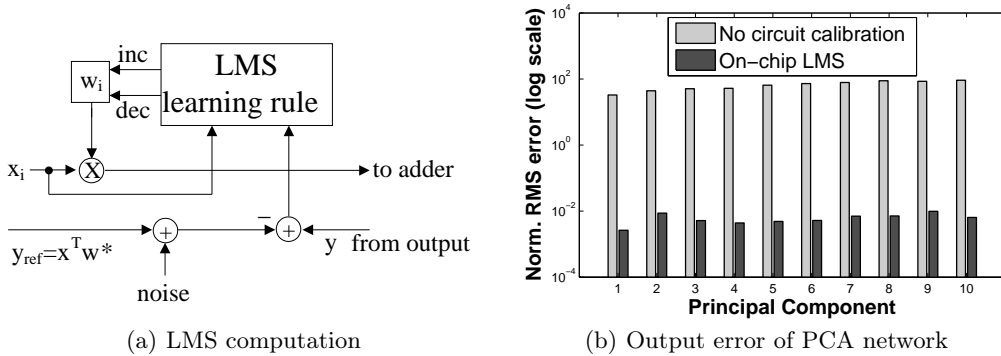

(a) LMS computation          (b) Output error of PCA network

Figure 3: Training the PCA network with LMS. (a) Block diagram of our LMS implementation. We present random inputs to each linear combiner, and provide a reference output. A pulse-based implementation of the LMS learning rule updates the memory cells. (b) RMS value of the error for the first 10 principal components, normalized to the RMS value of each PC.

where $\mu$ is the learning rate of the algorithm and $\mathbf{x}'_k$ is the reconstruction of the input $\mathbf{x}_k$ from the first principal component. The distortion introduced to the output by gain mismatch and offsets in Eqn. (9) is identical to Eqn. (6). Similarly to LMS, it is easy to show that a bias input $c$ connected to a synapse $b$ with an anti-Hebbian learning rule $\Delta b_k = \mu_b c y_k$ removes the constant offset added to the output. Therefore, we can eliminate the second term of Eqn. (6) and express the output as:

$$\overline{y}_k = \mathbf{x}_k^{\mathrm{T}}(\mathbf{A_x}\mathbf{A_w}\mathbf{w}_k + \mathbf{A_x}\boldsymbol{\gamma_w}) = \mathbf{x}_k^{\mathrm{T}}\overline{\mathbf{w}}_k \tag{12}$$

Using analog multipliers to compute $\mathbf{x}'_k$, we obtain:

$$\overline{\mathbf{x}'}_k = \overline{y}_k(\mathbf{A_y}\mathbf{A}'_\mathbf{w}\mathbf{w}_k + \mathbf{A_y}\boldsymbol{\gamma}'_\mathbf{w}) + \boldsymbol{\gamma_y}(\mathbf{A}'_\mathbf{w}\mathbf{w}_k + \boldsymbol{\gamma}'_\mathbf{w}) \tag{13}$$

where $\mathbf{A_y}$, $\mathbf{A}'_\mathbf{w}$, $\boldsymbol{\gamma_y}$, and $\boldsymbol{\gamma}'_\mathbf{w}$ are the gains and offsets associated with the multipliers used to compute $y_k\mathbf{w}_k$. Replacing Eqns. (12) and (13) in Eqn. (10), we determine the effective learning rule modified by device mismatch:

$$\overline{\Delta\mathbf{w}}_k = \mu\overline{y}_k(\mathbf{x} - \overline{y}_k(\mathbf{A_y}\mathbf{A}'_\mathbf{w}\mathbf{w}_k + \mathbf{A_y}\boldsymbol{\gamma}'_\mathbf{w})) = \mu\overline{y}_k(\mathbf{x} - \overline{y}_k\overline{\mathbf{w}}'_k) \tag{14}$$

If we use the same analog multipliers to compute $\overline{y}_k$ and $\overline{\mathbf{x}'}_k$, then $\mathbf{A_x} = \mathbf{A}_y$, $\mathbf{A_w} = \mathbf{A}'_\mathbf{w}$, and $\boldsymbol{\gamma_w} = \boldsymbol{\gamma}'_\mathbf{w}$, and the learning rule becomes:

$$\overline{\Delta\mathbf{w}}_k = \mu\overline{y}_k(\mathbf{x} - \overline{y}_k\overline{\mathbf{w}}_k) \tag{15}$$

where $\overline{y}_k$ and $\overline{\mathbf{w}}_k$ are the modified weight and output defined in Eqn. (12). Eqn. (15) is equivalent to the original learning rule in Eqn. (10), but with a new weight vector modified by device mismatch.

A convergence analysis for Eqn. (15) is complicated, but by analogy to LMS we can show that the weights indeed converge to the same values given in Eqns. (7) and (8), which compensate for the effects of gain mismatch and offset. Simulation results verify this assumption. Note that this will only be the case if we use the same hardware multipliers to compute $y_k$ and $\mathbf{x}'_k$. The analysis extends naturally to the higher-order principal components.

Fig. 4(a) shows our implementation of the GHA learning rule. The multiplexer shares the analog multipliers between the computation of $y_k$ and $\mathbf{x}'_k$, and is controlled by a digital signal that alternates its value during the computation and adaptation phases of the algorithm. Unlike LMS, GHA trains the algorithm using the images from the training set. Fig. 4(b) shows the normalized RMS value of the output error for the first 10 principal components. Comparing it to Fig. 3(b), the error is significantly higher than LMS, moving between 4% and 35% of the enery of the output. This higher error is due in part to the nonlinear multiply in the computation of $\mathbf{x}'_k$, and because there is a strong dependency between the learning rates used to update the bias synapse and the other weights in the network. However, as Section 5 shows, this error does not translate into a large degradation in the face classification performance.

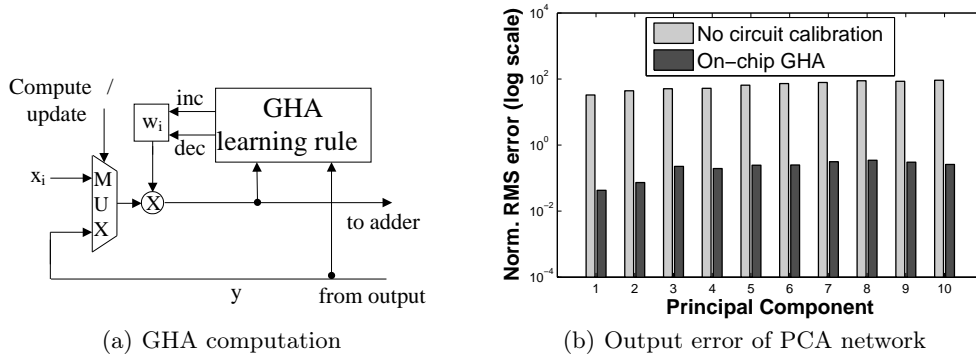

(a) GHA computation

(b) Output error of PCA network

Figure 4: Training the PCA network with GHA. (a) We reuse the multiplier to compute $\mathbf{x}'_k$ and use a pulse-based implementation of the GHA rule. (b) RMS value of the error for the first 10 principal components, normalized to the RMS value of each PC.

# 5   Classification Results

We designed and fabricated arithmetic circuits for the building blocks described in the previous sections using a $0.35\mu$m CMOS process, including analog memory cells, multipliers, and weight-update rules for LMS and GHA. We characterized these circuits in the lab and built a software emulator that allows us to test the static performance of different network configurations with less than 0.5% error. We simulated the LTA circuit in software. Using the emulator, we tested the performance of the face-recognition network on the Olivetti Research Labs (ORL) database, consisting on 10 photos of each of 40 total subjects. We used 5 random photos of each subject for the training set and 5 for testing. Limitations in our circuit emulator forced us to reduce the images to $12 \times 12$ pixels. The estimated power consumption of the circuit with these 144 inputs and 39 projections is 18mW (540nJ per classification with $30\mu$s settling time), and the layout area is 2.2mm$^2$. These numbers represent a 2–5x reduction in area and more than 100x reduction in power compated to standard cell-based digital implementations [4, 5].

Fig. 5(a) shows the classification performance of the network using PCA for dimensionality reduction, versus the number of principal components in the subspace. First, we tested the network using PCA for dimensionality reduction. The figure shows the performance of a software implementation of PCA with Euclidean distances, hardware PCA trained with LMS and software-computed weights, and hardware PCA trained with on-chip GHA. Both hardware implementations use Manhattan distances and a software LTA. The plots show the mean of the classification accuracy computed for each of the 40 individuals in the database. The error bars show one standard deviation above and below the mean. The software implementation peaks at 84% classification accuracy, while the hardware LMS and GHA implementations peak at 83% and 79%, respectively. Note that GHA performs only slightly worse than LMS, mainly because we compute and store the principal components of the training set in the face database using the same PCA network used to reduce the dimensionality of the test images, which helps to preserve the distance between classes in the feature space. The standard deviations are similar in all cases. Using an uncalibrated network brings the performance below 5%, mainly due to the offsets in the multipliers which change the PCA projection and take the signals outside of their nominal operating range.

Fig. 5(a) shows the classification results using the LDA in the dimensionality reduction network. The results are slightly better than PCA, and the error bars show also a lower variance. The performance of the software implementation of LDA and an a hard-classifier based on Euclidean distances is 83%. The LMS-trained hardware network with Manhattan distances and a software LTA yields 82%. Replacing the LTA with a software RBF classifier, the chip achieves 85% classification performance, while the software implementation (not shown) peaks at 87%. Using 40x40-pixel images and 39 projections, the software LDA network with RBF achieves more than 98% classification accuracy. Therefore, our current results are limited by the resolution of the input images.

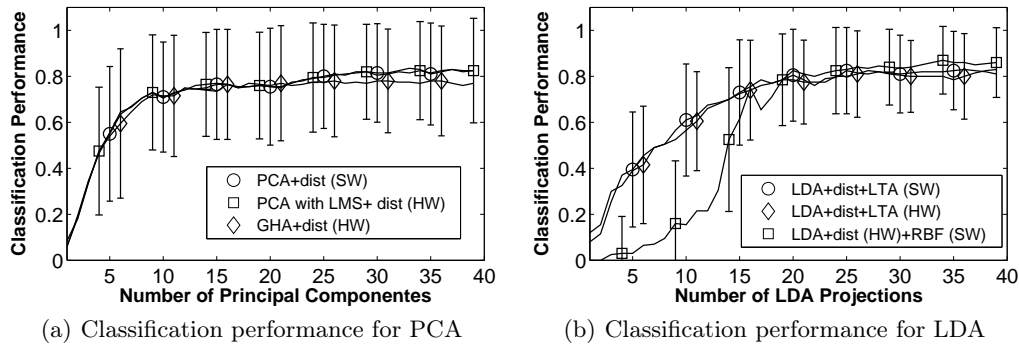

(a) Classification performance for PCA         (b) Classification performance for LDA

Figure 5: Classification performance for a $12 \times 12$–pixel version of the ORL database versus number of projections, using PCA and LDA for dimensionality reduction. Computing coefficients off-chip and writing them on the chip using LMS yields between 83% and 85% classification performance for PCA and LDA, respectively. This represents 98%-99% of the performance of a software implementation.

## 6    Conclusions

We presented an analog-VLSI network for face-recognition using subspace methods. We analyzed the effects of device mismatch on the performance of the dimensionality-reduction network and tested two techniques based on local adaptation which compensate for gain mismatch and offsets. We showed that using LMS to train the network on precomputed coefficients to perform PCA or LDA performs better than using GHA to learn PCA coefficients on chip. Ultimately, both techniques perform similarly in the face-classification task with the ORL database, achieving a classification performance of 83%-85% (98%-99% of a software implementation of the algorithms). Simulation results show that the performance is currently limited by the resolution of the input images. We are currently working on the integration LTA and RBF classifiers on chip, and on support of higher-dimensional inputs.

## Acknowledgments

This work was funded by the Chilean government through FONDECYT grant No. 1070485. The authors would like to thank Dr. Seth Bridges for his valuable contribution to this work.

## References

[1] M. Turk and A. Pentland. Face Recognition Using Eigenfaces. *Proc. of IEEE Conf. on Computer Vision and Pattern Recognition*, pages 586–591, 1991.

[2] Peter Belhumeur, Joao Hespanha, and David J. Kriegman. Eigenfaces vs. Fisherfaces: Recognition Using Class Specific Linear Projection". *IEEE Transactions on Pattern Analysis and Machine Intelligence*, 19(7):711–720, 1997.

[3] A. U. Batur, B. E. Flinchbaugh, and M. H. Hayes Ill. A DSP-Based approach for the implementation of face recognition algorithms. In *IEEE International Conference on Acoustics, Speech, and Signal Processing, 2003. Proceedings. (ICASSP '03)*, volume 2, pages 253–256, 2003.

[4] N. Shams, I. Hosseini, M. Sadri, and E. Azarnasab. Low Cost FPGA-Based Highly Accurate Face Recognition System Using Combined Wavelets Withs Subspace Methods. In *IEEE International Conference on Image Processing, 2006*, pages 2077–2080, 2006.

[5] C. S. S. Prasanna, N. Sudha, and V. Kamakoti. A Principal Component Neural Network-Based Face Recognition System and Its ASIC Implementation. In *VLSI Design*, pages 795–798, 2005.

[6] Ferdinando Samaria and Andy Harter. Parameterisation of a Stochastic Model for Human Face Identification. In *IEEE Workshop on Applications of Computer Vision*, Sarasota (Florida), December 1994.

[7] Miguel Figueroa, Esteban Matamala, Gonzalo Carvajal, and Seth Bridges. Adaptive Signal Processing in Mixed-Signal VLSI with Anti-Hebbian Learning. In *IEEE Computer Society Annual Symposium on VLSI*, pages 133–138, Karlsruhe, Germany, 2006. IEEE.

